# Learning Joint Statistical Models for Audio-Visual Fusion and Segregation

**John W. Fisher III**[*]
Massachusetts Institute of Technology
Cambridge, MA 02139
*fisher@ai.mit.edu*

**Trevor Darrell**
Massachusetts Institute of Technology
Cambridge, MA 02139
*trevor@ai.mit.edu*

**William T. Freeman**
Mitsubishi Electric Research Laboratory
Cambridge, MA 02139
*freeman@merl.com*

**Paul Viola**
Massachusetts Institute of Technology
Cambridge, MA 02139
*viola@ai.mit.edu*

## Abstract

People can understand complex auditory and visual information, often using one to disambiguate the other. Automated analysis, even at a low-level, faces severe challenges, including the lack of accurate statistical models for the signals, and their high-dimensionality and varied sampling rates. Previous approaches [6] assumed simple parametric models for the joint distribution which, while tractable, cannot capture the complex signal relationships. We learn the joint distribution of the visual and auditory signals using a non-parametric approach. First, we project the data into a maximally informative, low-dimensional subspace, suitable for density estimation. We then model the complicated stochastic relationships between the signals using a nonparametric density estimator. These learned densities allow processing across signal modalities. We demonstrate, on synthetic and real signals, localization in video of the face that is speaking in audio, and, conversely, audio enhancement of a particular speaker selected from the video.

## 1 Introduction

Multi-media signals pervade our environment. Humans face complex perception tasks in which ambiguous auditory and visual information must be combined in order to support accurate perception. By contrast, automated approaches for processing multi-media data sources lag far behind. Multi-media analysis (sometimes called sensor fusion) is often formulated in a maximum a posteriori (MAP) or maximum likelihood (ML) estimation framework. Simplifying assumptions about the joint measurement statistics are often made in order to yield tractable analytic forms. For example Hershey and Movellan have shown that correlations between video data and audio can be used to highlight regions of the image which are the "cause" of the audio signal. While such pragmatic choices may lead to

---

[*]http://www.ai.mit.edu/people/fisher

simple statistical measures, they do so at the cost of modeling capacity. Furthermore, these assumptions may not be appropriate for fusing modalities such as video and audio. The joint statistics for these and many other mixed modal signals are not well understood and are not well-modeled by simple densities such as multi-variate exponential distributions. For example, face motions and speech sounds are related in very complex ways.

A critical question is whether, in the absence of an adequate parametric model for joint measurement statistics, can one integrate measurements in a principled way without discounting statistical uncertainty. This suggests that a nonparametric statistical approach may be warranted. In the nonparametric statistical framework principles such as MAP and ML are equivalent to the information theoretic concepts of mutual information and entropy. Consequently we suggest an approach for learning maximally informative joint subspaces for multi-media signal analysis. The technique is a natural application of [8, 3, 5, 4] which formulates a learning approach by which the entropy, and by extension the mutual information, of a differentiable map may be optimized.

By way of illustration we present results of audio/video analysis using the suggested approach on both simulated and real data. In the experiments we are able to show significant audio signal enhancement and video source localization.

## 2    Information Preserving Transformations

Entropy is a useful statistical measure as it captures uncertainty in a general way. As the entropy of a density decreases so does the volume of the *typical* set [2]. Similarly, mutual information quantifies the information (uncertainty reduction) that two random variables convey about each other. The challenge of using such a measure for learning is that they are integral functions of densities (densities which must be inferred from samples).

### 2.1    Maximally Informative Subspaces

In order to make the problem tractable we project high dimensional audio and video measurements to low dimensional subspaces. The parameters of the sub-space are not chosen in an *ad hoc* fashion, but are learned by maximizing the mutual information between the derived features. Specifically, let $v_i \sim V \in \Re^{N_v}$ and $a_i \sim A \in \Re^{N_a}$ be video and audio measurements, respectively, taken at time $i$. Let $f_v : \Re^{N_v} \mapsto \Re^{M_v}$ and $f_a : \Re^{N_a} \mapsto \Re^{M_a}$ be mappings parameterized by the vectors $\alpha_v$ and $\alpha_a$, respectively. In our experiments $f_v$ and $f_a$ are single-layer perceptrons and $M_v = M_a = 1$. The method extends to any differentiable mapping and output dimensionality [3]. During adaptation the parameters vectors $\alpha_v$ and $\alpha_a$ (the perceptron weights) are chosen such that

$$\{\hat{\alpha}_v, \hat{\alpha}_a\} = \arg \max_{\alpha_v, \alpha_a} I(f_v(V, \alpha_v), f_a(A, \alpha_a)) \qquad (1)$$

This process is ilustrated notionally in figure 1 in which video frames and sequences of periodogram coefficients are projected to scalar values. A clear advantage of learning a projection is that rather than requiring pixels of the video frames or spectral coefficients to be inspected *individually* the projection summarizes the entire set efficiently into two scalar values (one for video and one for audio).

We have little reason to believe that joint audio/video measurements are accurately characterized by simple parametric models (e.g. exponential or uni-modal densities). Moreover, low dimensional projections which do not preserve this complex structure will not capture the true form of the relationship (i.e. random low dimensional projections of structured data are typically gaussian). The low dimensional projections which are learned by maximizing mutual information reduce the complexity of the joint distribution, but still preserve the important and potentially complex relationships between audio and visual signals. This

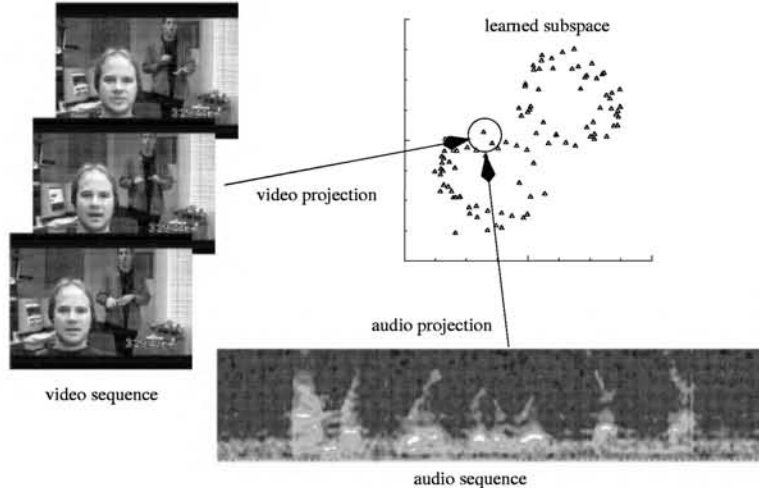

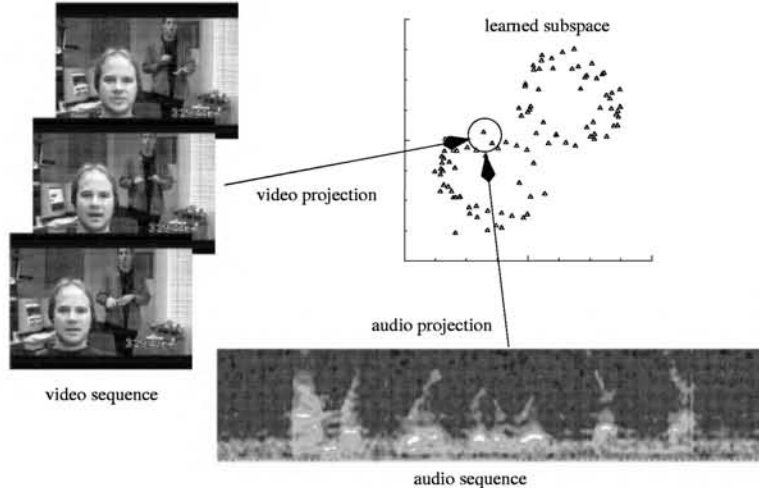

Figure 1: Fusion Figure: Projection to Subspace

possibility motivates the methodology of [3, 8] in which the density in the joint subspace is modeled nonparametrically.

This brings us to the natural question regarding the utitility of the learned supspace. There are a variety of ways the subspace and the associated joint density might be used to, for example, manipulate one of the disparate signals based on another. For the particular applications we address in our experiments we shall see that it is the mapping parameters $\{\hat{\alpha}_v, \hat{\alpha}_a\}$ which will be most useful. We will illustrate the details as we go through the experiments.

## 3 Empirical Results

In order to demonstrate the efficacy of the approach we present a series of audio/video analysis experiments of increasing complexity. In these experiments, two sub-space mappings are learned, one from video and another from audio.

In all cases, video data is sampled at 30 frames/second. We use both pixel based representations (raw pixel data) and motion based representations (i.e. optical flow [1]). Anandan's optical flow algorithm [1] is a coarse-to-fine method, implemented on a Laplacian pyramid, based on minimizing the sum of squared differences between frames. Confidence measures are derived from fitted quadratic surface principle curvatures. A smoothness constraint is also applied to the final velocity estimates. When raw video is used as an input to the subspace mapper, the pixels are collected into a single vector. The raw video images range in resolution from 240 by 180 (i.e.. 43,200 dimensions) to 320 by 240 (i.e. 76,800 dimensions). When optical flow is used as an input to the sub-space mapper, vector valued flow for each pixel is collected into a single vector, yielding an input vector with twice as many dimensions as pixels.

Audio data is sampled at 11.025 KHz. Raw audio is transformed into periodogram coefficients. Periodograms are computed using hamming windows of 5.4 ms duration sampled at 30 Hz (commensurate with the video rate). At each point in time there are 513 periodogram coefficients input to the sub-space mapper.

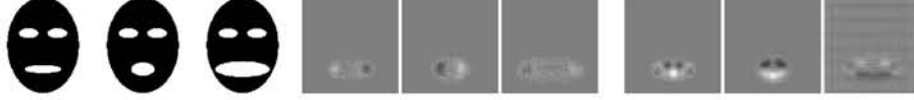

Figure 2: Synthetic image sequence examples (left). Mouth parameters are functionally related to one audio signal. Flow fields horizontal component (center) and vertical component (right).

### 3.1 A Simple Synthetic Example

We begin with a simple synthetic example. The goal of the experiment is to use a video sequence to enhance an associated audio sequence. Figure 2 shows examples from a synthetically generated image sequence of faces (and the associated optical flow field). In the sequence the mouth is described by an ellipse. The parameters of the ellipse are functionally related to a recorded audio signal. Specifically, the area of the ellipse is proportional to the average power of the audio signal (computed over the same periodogram window) while the eccentricity is controlled by the the entropy of the normalized periodogram. Consequently, observed changes in the image sequence are functionally related to the recorded audio signal. It is not necessary (right now) that the relationship be realistic, only that it exists. The associated audio signal is mixed with an interfering, or noise, signal. Their spectra, shown in figure 3 (left), are clearly overlapped.

If the power spectrum of the associated and interfering signals were known then the optimal filter for recovering the associated audio sequence is the Wiener filter. It's spectrum is described by

$$H(f) = \frac{P_a(f)}{P_a(f) + P_n(f)} \tag{2}$$

where $P_a(f)$ is the power spectrum of the desired signal and $P_n(f)$ is the power spectrum of the interfering signal. In general this information is unknown, but for our experiments it is useful as a benchmark for comparison purposes as it represents an upper bound on performance. That is, in a second-order sense, all filters (including ours) will underperform the Wiener filter. Furthermore, suppose $y = s_a + n$ where $s_a$ is the signal of interest and $n$ is an independent interference signal. It can be shown that

$$\rho^2 = \sqrt{\frac{\left(\frac{s}{n}\right)}{1 + \left(\frac{s}{n}\right)}} \quad \leftrightarrow \quad \left(\frac{s}{n}\right) = \frac{\rho^2}{1 - \rho^2} \tag{3}$$

where $\rho$ is the correlation coefficient between $s_a$ and the corrupted version $y$ and $\left(\frac{s}{n}\right)$ is the signal to noise power ratio (SNR). Consequently given a reference signal and some signal plus interferer we can use the relationships above to gauge signal enhancement.

The question we address is that in the absence of knowing the separate power spectra, which are necessary to implement the Wiener filter, how do we compare using the associated video data. It is not immediately obvious how one might achieve signal enhancement by learning a joint subspace in the manner described. Our intuition is as follows. For this simple case it is only the associated audio signal which bears any relationship to the video sequence. Furthermore, the coefficients of the audio projection, $\alpha_a$ correspond to spectral coefficients. Our reasoning is that large magnitude coefficients correspond those spectral components which have more signal component than those with small magnitude. Using this reasoning we can construct a filter whose coefficients are proportional to our projection $\alpha_a$. Specifically, we use the following to design our filter

$$H_{MI}(f) = \beta \left( \frac{|\alpha_a(f)| - \min(|\alpha_a(f)|)}{\max(|\alpha_a(f)|) - \min(|\alpha_a(f)|)} \right) + \frac{1 - \beta}{2} \; ; \; 0 \le \beta \le 1 \tag{4}$$

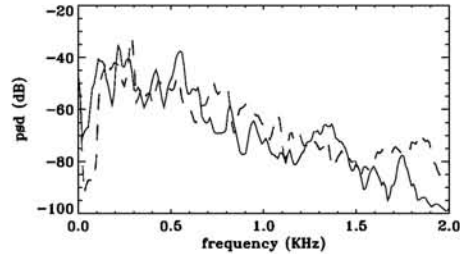

Figure 3: Spectra of audio signals (right). Solid line indicates the desired audio component while the dashed line indicates the interference.

where $\alpha_a(f)$ are the audio projection coefficients associated with spectral coeffiecient, $f$. For our experiements, $\beta = 0.90$, consequently $0.5 \leq H_{MI}(f) \leq 0.95$. While somewhat *ad hoc* the filter is consistent with our reasoning above and, as we shall see, yields good results.

Furthermore, because the signal and interferer are known (in our experimental set up) we can compare our results to the unachievable, yet optimal, Wiener filter for this case. In this case the SNR was 0 dB, furthermore as the two signals have significant spectral overlap, signal recovery is challenging. The optimal Wiener filter achieves a signal processing gain of 2.6 dB while the filter constructed as described achieves 2.0 dB (when using images directly) and 2.1 db when using optical flow.

### 3.2 Video Attribution of Single Audio Source

The previous example demonstrated that the audio projection coefficients could be used to reduce an interfering signal. We move now to a different experiment using real data. Figure 4(a) shows a video frame from the sequence used in the next experiment. In the scene there is a person speaking in the foreground, a person moving in the background and a monitor which is flickering. There is a single audio signal source (of the speaker) but several interfering motion fields in the video sequence. Figures 4(b) is the pixel-wise standard deviations of the video sequence while figure 4(c) shows the pixel-wise flow field energy. These images show that there are many sources of change in the image. Note that the most intense changes in the image sequence are associated with the monitor and not the speaker. Our goal with this experiment is to show that via the method described we can properly attribute the region of the video image which is associated with the audio sequence. The intuition is similar to the previous experiment. We expect that large image projection coefficients, $\alpha_v$ correspond to those pixels which are related to the audio signal. Figure 4(d) shows the image $\alpha_v$ when images are fed directly into the algorithm while figure 4(e) shows the same image when flow-fields are the input. Clearly both cases have detected regions associated with the speaker with the substantive difference being that the use of flow fields resulted in a smoother attribution.

### 3.3 User-assisted Audio Enhancement

We now repeat the initial synthetic experiment of 3.1 using real data. In this case there are two speakers recorded with a single microphone (the speakers were recorded with stereo microphones so as to obtain a reference, but the experiments used a single mixed audio source). Figure 5(a) shows an example frame from the video sequence. We now demonstrate the ability to enhance the audio signal in a user-assisted fashion. By selecting data

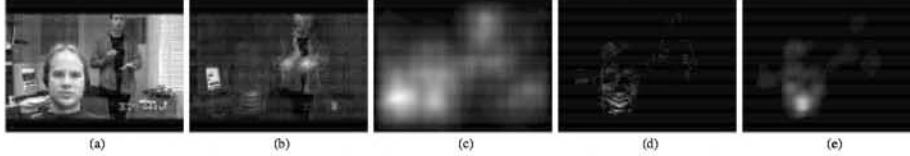

Figure 4: Video attribution: (a) example image, (b) pixel standard deviations, (c) flow vector energy, (d) image of $\alpha_v$ (pixel features), (e) flow field features

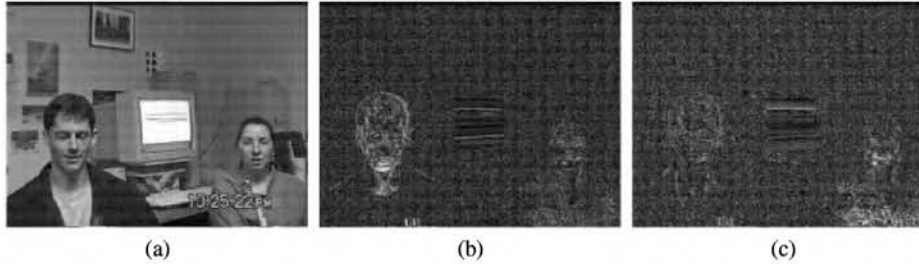

Figure 5: User assisted audio enhancement: (a) example image, with user chosen regions, (b) image of $\alpha_v$ for region 1, (c) image of $\alpha_v$ region 2

from one box or the other in figure 5(a) we can enhance the voice of the speaker on the left or right. As the original data was collected with stereo microphones we can again compare our result to an approximation to the Wiener filter (neglecting cross channel leakage). In this case, due to the fact that the speakers are male and female, the signals have better spectral separation. Consequently the Wiener filter achieves a better signal processing gain. For the male speaker the Wiener filter improves the SNR by 10.43 dB, while for the female speaker the improvement is 10.5 dB. Using our technique we are able to achieve a 8.9 dB SNR gain (pixel based) and 9.2 dB SNR gain (optic flow based) for the male speaker while for the female speaker we achieve 5.7 and 5.6 dB, respectively.

It is not clear why performance is not as good for the female speaker, but figures 5(b) and (c) are provided by way of partial explanation. Having recovered the audio in the user-assisted fashion described we used the recovered audio signal for video attribution (pixel-based) of the entire scene. Figures 5(b) and (c) are the images of the resulting $\alpha_v$ when using the male (b) and female (c) recovered voice signals. The attribution of the male speaker in (b) appears to be clearer than that of (c). This may be an indication that the video cues were not as detectable for the female speaker as they were for the male in this experiment. In any event these results are consistent with the enhancement results described above.

## 4 Applications

There are several practical applications for the techniques described in this paper. One key area is speech recognition. Recent commercial advances in speech recognition rely on careful placement of the microphone so that background sounds are minimized. Results in more natural environments, where the microphone is some distance from the speaker and there is significant background noise, are disappointing. Our approach may prove useful for teleconferencing, where audio and video of multiple speakers is recorded simultaneously.

Other applications include broadcast television in situations where careful microphone placement is not possible, or post-hoc processing to enhance the audio channel might prove

valuable. For example, if one speaker's microphone at a news conference malfunctions, the voice of that speaker might be enhanced with the aid of video information.

## 5  Conclusions

One key contribution of this paper is to extend the notion of multi-media fusion to complex domains in which the statistical relationships between audio and video is complex and non-gaussian. This is claim is supported in part by the results of Slaney and Covell in which canonical correlations failed to detect audio/video synchrony when a spectral representation was used for the audio signal [7]. Previous approaches have attempted to model these relationships using simple models such as measuring the short term correlation between pixel values and the sound signal [6]. The power of the non-parametric mutual information approach allows our technique to handle complex non-linear relationships between audio and video signals. One demonstration of this modeling flexibility, is the insensitivity to the form of the input signals. Experiments were performed using raw pixel intensities as well as optical flows (which is a complex non-linear function of pixel values across time), yielding similar results.

Another key contribution is to establish an important application for this approach, video enhanced audio segmentation. Initial experiments have shown that information from the video signal can be used to reduce the noise in a simultaneously recorded audio signal. Noise is reduced without any a priori information about the form of the audio signal or noise. Surprisingly, in our limited experiments, the noise reduction approaches what is possible using a priori knowledge of the audio signal (using Weiner filtering).

## References

[1] P. Anandan. A computational framework and an algorithm for the measurement of visual motion. *Int. J. comp. Vision*, 2:283–310, 1989.

[2] T. M. Cover and J. A. Thomas. *Elements of Information Theory*. John Wiley & Sons, Inc., New York, 1991.

[3] J. Fisher and J. Principe. Unsupervised learning for nonlinear synthetic discriminant functions. In D. Casasent and T. Chao, editors, *Proc. SPIE, Optical Pattern Recognition VII*, volume 2752, pages 2–13, 1996.

[4] J. W. Fisher III, A. T. Ihler, and P. A. Viola. Learning informative statistics: A nonparametric approach. In S. A. Solla, T. K. Leen, and K.-R. Mller, editors, *Proceedings of 1999 Conference on Advances in Neural Information Processing Systems 12*, 1999.

[5] J. W. Fisher III and J. C. Principe. A methodology for information theoretic feature extraction. In A. Stuberud, editor, *Proceedings of the IEEE International Joint Conference on Neural Networks*, 1998.

[6] J. Hershey and J. Movellan. Using audio-visual synchrony to locate sounds. In T. K. L. S. A. Solla and K.-R. Mller, editors, *Proceedings of 1999 Conference on Advances in Neural Information Processing Systems 12*, 1999.

[7] M. Slaney and M. Covell. Facesync: A linear operator for measuring synchronization of video facial images and audio tracks. In *This volume*, 2001.

[8] P. Viola, N. Schraudolph, and T. Sejnowski. Empirical entropy manipulation for real-world problems. In *Proceedings of 1996 Conference on Advances in Neural Information Processing Systems 8*, pages 851–7, 1996.
